# The Role of MT Neuron Receptive Field Surrounds in Computing Object Shape from Velocity Fields

G.T.Buracas & T.D.Albright
Vision Center Laboratory, The Salk Institute,
P.O.Box 85800, San Diego, California 92138-9216

## Abstract

The goal of this work was to investigate the role of primate MT neurons in solving the structure from motion (SFM) problem. Three types of receptive field (RF) surrounds found in area MT neurons (K.Tanaka *et al.*,1986; Allman *et al.*,1985) correspond, as our analysis suggests, to the $0^{th}$, $1^{st}$ and $2^{nd}$ order fuzzy space-differential operators. The large surround/center radius ratio ($\geq$ 7) allows both differentiation of smooth velocity fields and discontinuity detection at boundaries of objects. The model is in agreement with recent psychophysical data on surface interpolation involvement in SFM. We suggest that area MT partially segregates information about object shape from information about spatial relations necessary for navigation and manipulation.

## 1 INTRODUCTION

Both neurophysiological investigations [8] and lesioned human patients' data show that the Middle Temporal (MT) cortical area is crucial to perceiving three-dimensional shape in moving stimuli. On the other hand,

a solid body of data (e.g. [1]) has been gathered about functional properties of neurons in the area MT. Hoever, the relation between our ability to perceive structure in stimuli, simulating 3-D objects, and neuronal properties has not been addressed up to date. Here we discuss a possibility, that area MT RF surrounds might be involved in shape-from-motion perception. We introduce a simplifying model of MT neurons and analyse the implications to SFM problem solving.

## 2 REDEFINING THE SFM PROBLEM

### 2.1 RELATIVE MOTION AS A CUE FOR RELATIVE DEPTH

Since Helmholtz motion parallax is known to be a powerful cue providing information about both the structure of the surrounding environment and the direction of self-motion. On the other hand, moving objects also induce velocity fields allowing judgement about their shapes. We can capture both cases by assuming that an observer is tracking a point on a surface of interest. The velocity field of an object then is (fig.1): $V = t_z + w \times (R - R_0)$ $= -t_z + w \times z$, where $w = [w_x, w_y, 0]$ is an effective rotation vector of a surface $z = [x, y, z(x,y)]$; $R_0 = [0, 0, z_0]$ is a positional vector of the fixation point; $t_z$ is a translational component along Z axis.

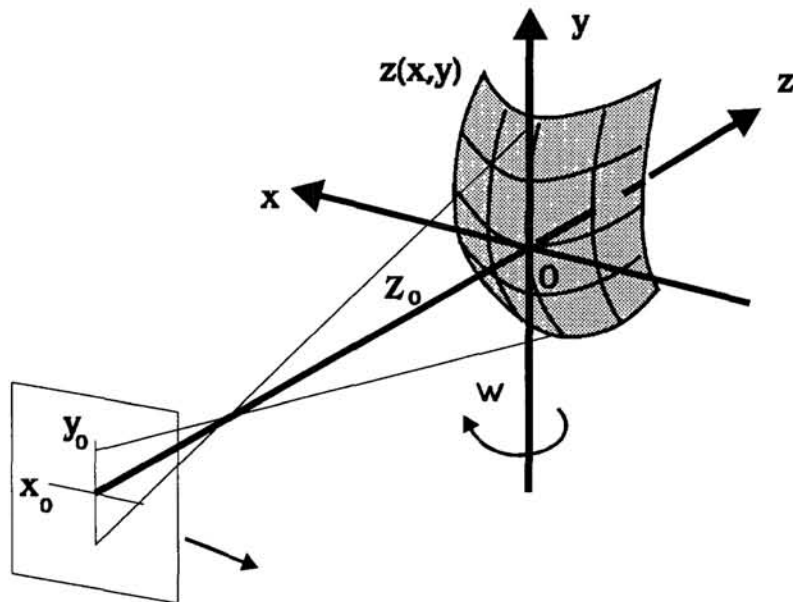

Fig.1: The coordinate system assumed in this paper. The origin is set at the fixation point. The observer is at $Z_0$ distance from a surface.

The component velocities of a retinal velocity field under perspective projection can be calculated from:

$$u = -\frac{w_y z}{z_0 + z} - \frac{-xt_z - w_x xy + w_y x^2}{(z_0 + z)^2}, \qquad v = \frac{w_x z}{z_0 + z} - \frac{-yt_z + w_y xy - w_x y^2}{(z_0 + z)^2}.$$

In natural viewing conditions the distance to the surface $z_0$ is usually much larger than variation in distance on the surface $z : z_0 \gg z$. In such the second term in the above equations vanishes. In the case of translation tangential to the ground, to which we confine our analysis, $\mathbf{w}=[0,w_y,0] = [0,w,0]$, and the retinal velocity reduces to

$$u = -wz/(z_0+z) \approx -wz/z_0 , \qquad v=0 \qquad (1).$$

The latter relation allows the assumption of orthographic projection, which approximates the retinal velocity field rather well within the central 20 deg of the visual field.

## 2.2 SFM PERCEPTION INVOLVES SURFACE INTERPOLATION

Human SFM perception is characterized by an interesting peculiarity -- surface interpolation [7]. This fact supports the hypothesis that an assumption of surface continuity is embedded in visual system. Thus, we can redefine the SFM problem as a problem of characterizing the interpolating surfaces. The principal normal curvatures are a local measure of surface invariant with respect to translation and rotation of the coordinate system. The orientation of the surface (normal vector) and its distance to the observer provide the information essential for navigation and object manipulation. The first and second order differentials of a surface function allow recovery of both surface curvature and orientation.

# 3  MODEL OF AREA MT RECEPTIVE FIELD SURROUNDS

## 3.1  THREE TYPES OF RECEPTIVE FIELD SURROUNDS

The Middle Temporal (MT) area of monkeys is specialized for the systematic representation of direction and velocity of visual motion [1,2]. MT neurons are known to posess large, *silent* (RFS, the "nonclassical RF". Born and Tootell [4] have very recently reported that the RF surrounds of neurons in owl monkey MT can be divided into antagonistic and *synergistic* types (Fig.2a).

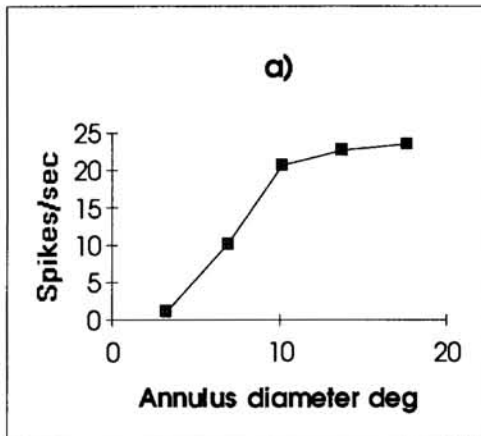

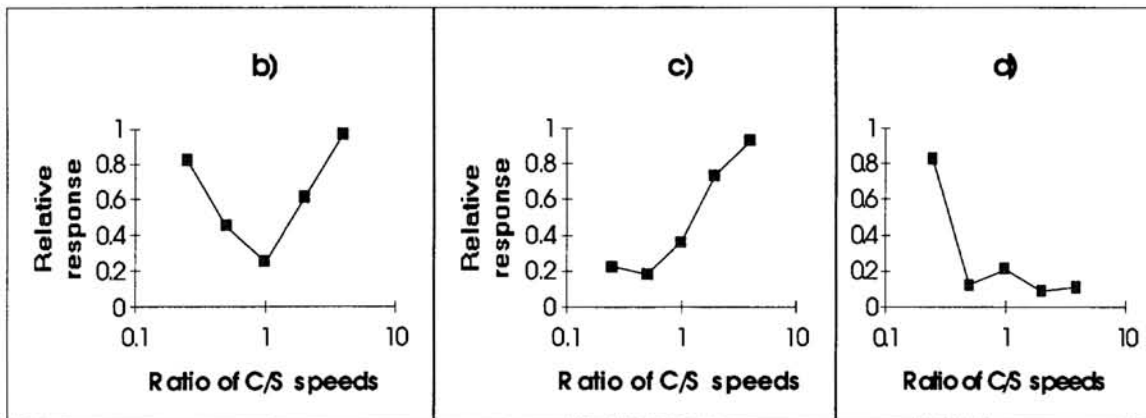

Fig.2: Top left (a): an example of a synergistic RF surround, redrawn from [4] (no velocity tuning known). Bottom left (b): a typical V-shaped tuning curve for RF surround The horizontal axis represents the logarithmic scale of ratio between stimulus speeds in the RF center and surround, redrawn from [9]. Bottom (c,d): monotonically increasing and decreasing tuning curves for RF surrounds, redrawn from [9].

About 44% of the owl monkey neuron RFSs recorded by Allman *et al.* [3] showed *antagonistic* properties. Approximately 33% of these demonstrated *V(or U)-shaped* (Fig.2b), and 66% - *quasi-linear* velocity tuning curves (Fig.2c,d). One half of *Macaca fuscata* neurons with antagonistic RFS found by Tanaka *et al* [9] have had V(U)-shaped velocity tuning curves, and 50% monotonically increasing or decreasing velocity tuning curves. The RFS were tested for symmetry [9] and *no asymmetrical surrounds were found* in primate MT.

### 3.2 CONSTRUCTING IDEALIZED MT FILTERS

The surround (S) and center (C) responses seem to be largely independent (except for the requirement that the velocity in the center must be nonzero) and seem to combine in an additive fashion [5]. This property allows us to combine C and S components in our model independently. The resulting filters can be reduced to three types, described below.

### 3.2.1 Discrete Filters

The essential properties of the three types of RFSs in area MT can be captured by the following difference equations. We choose the slopes of velocity tuning curves in the center to be equal to the ones in the surround; this is essential for obtaining the desired properties for $l_2$ but not $l_0$. The 0-order (or low-pass) and the 2nd order (or band-pass) filters are defined by:

$$l_0 = g_0 \sum_i \sum_j (u_c + u_s(i,j)) + Const0, ; \quad l_2 = g_2 \sum_i \sum_j (u_c - u_s(i,j)) + Const2, \quad (2)$$

where g is gain, $w_{i,j} = 1$, $i,j \in [-r,r]$ (r = radius of integration). Speed scalars $u(i,j)$ at points [i,j] replace the velocity vectors $V$ due to eq. (1). Constants correspond to spontaneous activity levels.

In order to achieve the V(U) -shaped tuning for the surround in Fig.2b, a nonlinearity has to be introduced:

$$l_1 = g_1 \sum_i \sum_j (u_c - u_s(i,j))^2 + Const1. \quad (3)$$

The responses of $l_1$ and $l_2$ filters to standard mapping stimuli used in [3,9] are plotted together with their biological correlates in Fig.3.

### 3.2.2 Continuous analogues of MT filters

We now develop continuous, more biologicaly plausible, versions of our three MT filters. We assume that synaptic weights for both center and surround regions fall off with distance from the RF center as a Gaussian function G(x,y,σ), and σ is different for center and surround: $\sigma_c \neq \sigma_s$. Then, by convolving with Gaussians equation (2) can be rewritten:

$$L_0(i,j) = u(i,j)*G(\sigma_c) + u(i,j)*G(\sigma_s),$$

$$L_2^{\pm}(i,j) = \pm[u(i,j)*G(\sigma_c) - u(i,j)*G(\sigma_s)].$$

The continuous nonlinear $L_1$ filter can be defined if equivalence to $l_1$ (eq. 3) is observed only up to the second order term of power series for u(i,j):

$$L_1(i,j) = u^2(i,j)*G(\sigma_c) + u^2(i,j)*G(\sigma_s) - C \cdot [u(i,j)*G(\sigma_c)] \cdot [u(i,j)*G(\sigma_s)];$$

$u^2(i,j)$ corresponds to full-wave rectification and seems to be common in area V1 complex neurons; $C = 2/Erf^2(n/2^{1/2})$ is a constant, and Erf() is an error function.

### 3.3 THE ROLE OF MT NEURONS IN SFM PERCEPTION.

Expanding z(x,y) function in (1) into power series around an arbitrary point and truncating above the second order term yields: $u(x,y)=w(ax^2+by^2+cxy+dx+ey+f)/z_0$, where a,b,c,d,e,f are expansion coefficients. We assume that w is known (from proprioceptive input) and =1. Then $z_0$ remans an unresolved scaling factor and we omit it for simplicity.

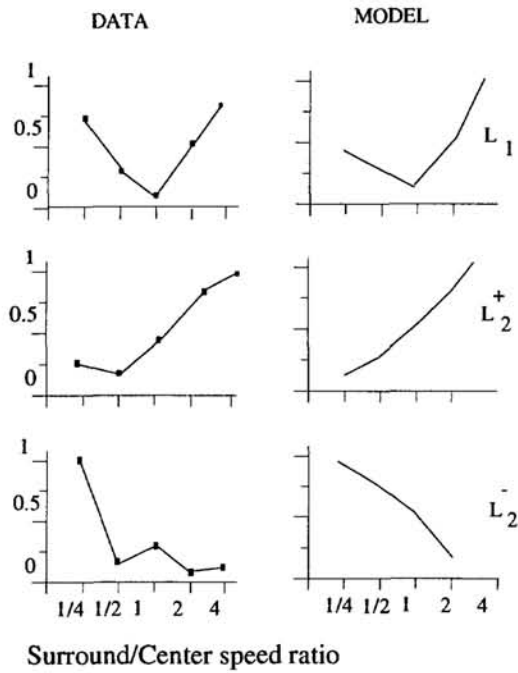

DATA    MODEL

$L_1$

$L_2^+$

$L_2^-$

1/4 1/2 1  2  4      1/4 1/2 1  2  4

Surround/Center speed ratio

Fig. 3: The comparison between data [9] and model velocity tuning curves for RF surrounds. The standard mapping stimuli (optimaly moving bar in the center of RF, an annulus of random dots with varying speed) were applied to $L_1$ and $L_2$ filters. Thee output of the filters was passed through a sigmoid transfer function to accout for a logarithmic compresion in the data.

Fig. 4: Below, left: the response profile of the $L_1$ filter in orientation space (x and y axes represent the components of normal vector ). Right: the response profile of the $L_2$ filter in curvature space. x and y axes represent the two normal principal curvatures.

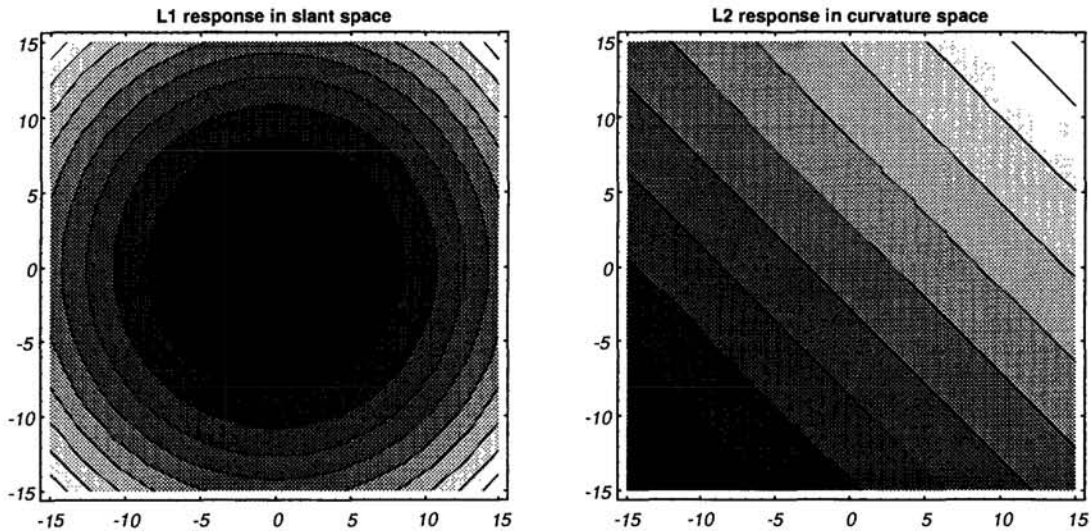

Applying $L_0$ on u(x,y), high spatial frequency information is filtered out, but otherwise u(x,y) does not change, i.e. $L_0*u$ covaries with lower frequencies of u(x,y). $L_2$ applied on u(x,y) yields:

$$L_2 * u = (2a + 2b)C_2(\sigma_c^2 - \sigma_s^2) = C_2(\sigma_c^2 - \sigma_s^2)\nabla^2 u, \quad (4)$$

that is, $L_2$ shows properties of the second order space-differential operator - Laplacian; $C_2(\sigma_c^2 - \sigma_s^2)$ is a constant depending only on the widths of the center and surround Gaussians. Note that $L_2*u \cong \kappa_1 + \kappa_2$, ($\kappa_{1,2}$ are principal normal curvatures) at singular points of surface z(x,y).

When applied on planar stimuli $u_p(x,y) = d\,x + e\,y$, $L_1$ has properties of a squared first order differential operator:

$$L_1 * u_p = (d^2 + e^2) C_1 (\sigma_c^2 - \sigma_s^2) = C_1 (\sigma_c^2 - \sigma_s^2) \left( (\frac{\partial}{\partial x})^2 + (\frac{\partial}{\partial y})^2 \right) u_p, \quad (5)$$

where $C_2(\sigma_c^2 - \sigma_s^2)$ is a function of $\sigma_c$ and $\sigma_s$ only. Thus the output of $L_1$ is monotonically related to the norm of gradient vector. It is straightforward to calculate the generic second order surface based on outputs of three $L_0$, four $L_1$ and one $L_2$ filters.

Plotting the responses of $L_1$ and $L_2$ filters in orientation and curvature space can help to estimate the role they play in solving the SFM problem (Fig.4). The iso-response lines in the plot reflect the ambiguity of MT filter responses. However, these responses covary with useful geometric properties of surfaces -- norm of gradient ($L_1$) and mean curvature ($L_2$).

### 3.4 EXTRACTING VECTOR QUANTITIES

Equations (4) and (5) show, that only averaged scalar quantities can be extracted by our MT operators. The second order directional derivatives for estimating vectorial quantities can be computed using an oriented RFs with the following profile: $O_2 = G(x,\sigma_s) [G(y,\sigma_s) - G(y,\sigma_c)]$. $O_1$ then can be defined by the center - surround relationship of $L_1$ filter. The outputs of MT filters $L_1$ and $L_2$ might be indispensible in normalizing responses of oriented filters. The normal surface curvature can be readily extracted using combinations of MT and hypothetical O filters. The oriented spatial differential operators have not been found in primate area MT so far. However, preliminary data from our lab indicate that elongated RFs may be present in areas FST or MST [6].

### 3.5 $L_2$: LAPLACIAN VS. NAKAYAMA'S CONVEXITY OPERATOR

The physiologically tested ratio of standard deviations for center and surround Gaussians $\sigma_s/\sigma_c \geq 7$. Thus, besides performing the second order differentiation in the low frequency domain, $L_2$ can detect discontinuities in optic flow.

## 4. CONCLUSIONS

We propose that the RF surrounds in MT may enable the neurons to function as differential operators. The described operators can be thought of as providing a continuous interpolation of cortically represented surfaces.

Our model predicts that elongated RFs with flanking surrounds will be found (possibly in areas FST or MST [6]). These RFs would allow extraction

of the directional derivatives necessary to estimate the principal curvatures and the normal vector of surfaces.

From velocity fields, area MT extracts information relevant to both the "where" stream (motion trajectory, spatial orientation and relative distance of surfaces) and the "what" stream (curvature of surfaces).

## Acknowledgements

Many thanks to George Carman, Lisa Croner, and Kechen Zhang for stimulating discussions and Jurate Bausyte for helpful comments on the poster. This project was sponsored by a grant from the National Eye Institute to TDA and by a scholarship from the Lithuanian Foundation to GTB. The presentation was supported by a travel grant from the NIPS foundation.

## References

[1] Albright, T.D. (1984) Direction and orientation selectivity of neurons in visual area MT of the macaque. *J. Neurophysiol.*, **52**: 1106-1130.

[2] Albright, T.D., R.Desimone. (1987) Local precision of visuotopic organization in the middle temporal area (MT) of the macaque. *Exp.Brain Res.*, **65**, 582-592.

[3] Allman, J., Miezin, F., McGuinnes. (1985) Stimulus specific responses from beyond the classical receptive field. Ann.Rev.Neurosci., **8**, 407-430.

[4] Born R.T. & Tootell R.B.H. (1992) Segregation of global and local motion processing in primate middle temporal visual area. *Nature*, **357**, 497-499.

[5] Born R.T. & Tootell R.B.H. (1993) Center - surround interactions in direction - selective neurons of primate visual area MT. *Neurosci. Abstr.*, **19**, 315.5.

[6] Carman G.J., unpublished results.

[7] Hussain M., Treue S. & Andersen R.A. (1989) Surface interpolation in three-dimensional Structure-from-Motion perception. *Neural Computation*, 1, 324-333.

[8] Siegel, R.M. and R.A. Andersen. (1987) Motion perceptual deficits following ibotenic acid lesions of the middle temporal area in the behaving rhesus monkey. *Soc.Neurosci.Abstr.*, **12**, 1183.

[9]Tanaka, K., Hikosaka, K., Saito, H.-A., Yukie, M., Fukada, Y., Iwai, E. (1986) Analysis of local and wide-field movements in the superior temporal visual areas of the macaque monkey. *J.Neurosci.*, **6**, 134-144.